# A Sparse Non-Parametric Approach for Single Channel Separation of Known Sounds

**Paris Smaragdis**
Adobe Systems Inc.
paris@adobe.com

**Madhusudana Shashanka**
Mars Inc.
shashanka@alum.bu.edu

**Bhiksha Raj**
Carnegie Mellon University
bhiksha@cs.cmu.edu

## Abstract

In this paper we present an algorithm for separating mixed sounds from a monophonic recording. Our approach makes use of training data which allows us to learn representations of the types of sounds that compose the mixture. In contrast to popular methods that attempt to extract compact generalizable models for each sound from training data, we employ the training data itself as a representation of the sources in the mixture. We show that mixtures of known sounds can be described as sparse combinations of the training data itself, and in doing so produce significantly better separation results as compared to similar systems based on compact statistical models.

**Keywords:** Example-Based Representation, Signal Separation, Sparse Models.

## 1 Introduction

This paper deals with the problem of single-channel signal separation – separating out signals from individual sources in a mixed recording. As of recently, a popular statistical approach has been to obtain compact characterizations of individual sources and employ them to identify and extract their counterpart components from mixture signals. Statistical characterizations may include codebooks [1], Gaussian mixture densities [2], HMMs [3], independent components [4, 5], sparse dictionaries [6], non-negative decompositions [7–9] and latent variable models [10, 11]. All of these methods attempt to derive a generalizable model that captures the salient characteristics of each source. Separation is achieved by abstracting components from the mixed signal that conform to the statistical characterizations of the individual sources. The key here is the specific statistical model employed – the more effectively it captures the specific characteristics of the signal sources, the better the separation that may be achieved.

In this paper we argue that, given any sufficiently large collection of data from a source, the best possible characterization of any data is, quite simply, the data themselves. This has been the basis of several example-based characterizations of a data source, such as nearest-neighbor, K-nearest neighbor, Parzen-window based models of source distributions etc. Here, we use the same idea to develop a monaural source-separation algorithm that directly uses samples from the training data to represent the sources in a mixture. Using this approach we sidestep the need for a model training step, and we can rely on a very flexible reconstruction process, especially as compared with previously used statistical models. Identifying the proper samples from the training data that best approximate a sample of the mixture is of course a hard combinatorial problem, which can be computationally demanding. We therefore formulate this as a sparse approximation problem and proceed to solve it with an efficient algorithm. We additionally show that this approach results in

source estimates which are guaranteed to lie on the source manifold, as opposed to trained-basis approaches which can produce arbitrary outputs that will not necessarily be plausible source estimates.

Experimental evaluations show that this approach results in separated signals that exhibit significantly higher performance metrics as compared to conceptually similar techniques which are based on various types of combinations of generalizable bases representing the sources.

## 2 Proposed Method

In this section we cover the underlying statistical model we will use, introduce some of the complications that one might encounter when using it and finally we propose an algorithm that resolves these issues.

### 2.1 The Basic Model

Given a magnitude spectrogram of a single source, each spectral frame is modeled as a histogram of repeated draws from a multinomial distribution over the frequency bins. At a given time frame $t$, consider a random process characterized by the probability $P_t(f)$ of drawing frequency $f$ in a given draw. The distribution $P_t(f)$ is unknown but what one can observe instead is the result of multiple draws from the process, that is the observed spectral vector. The model assumes that $P_t(f)$ is comprised of bases indexed by a latent variable $z$. The latent factors are represented by $P(f|z)$. The probability of picking the $z$-th distribution in the $t$-th time frame can be represented by $P_t(z)$. We use this model to learn the source-specific bases given by $P_t(f|z)$ as done in [10,11]. At this point this model is conceptually very similar to the non-negative factorization models in [8,9].

Now let the matrix $\mathbf{V}_{F \times T}$ of entries $v_{ft}$ represent the magnitude spectrogram of the mixture sound and $\mathbf{v}_t$ represent time frame $t$ (the $t$-th column vector of matrix $\mathbf{V}$). Each mixture spectral frame is again modeled as a histogram of repeated draws, from the multinomial distributions corresponding to every source. The model for each mixture frame includes an additional latent variable $s$ representing each source, and is given by

$$P_t(f) = \sum_s P_t(s) \sum_{z \in \{\mathbf{z}_s\}} P_s(f|z)P_t(z|s), \tag{1}$$

where $P_t(f)$ is the probability of observing frequency $f$ in time frame $t$ in the mixture spectrogram, $P_s(f|z)$ is the probability of frequency $f$ in the $z$-th learned basis vector from source $s$, $P_t(z|s)$ is the probability of observing the $z$-th basis vector of source $s$ at time $t$, $\{\mathbf{z}_s\}$ represents the set of values the latent variable $z$ can take for source $s$, and $P_t(s)$ is the probability of observing source $s$ at time $t$.

We can assume that for each source in the mixture we have an already trained model in the form of basis vectors $P_s(f|z)$. These bases will represent a dictionary of spectra that best describe each source. Armed with this knowledge we can decompose a new mixture of these known sources in terms of the contributions of the dictionaries for each source. To do so we can use the EM algorithm to estimate $P_t(z|s)$ and $P_t(s)$:

$$P_t(s, z|f) = \frac{P_t(s)P_t(z|s)P_s(f|z)}{\sum_s P_t(s) \sum_{z \in \{\mathbf{z}_s\}} P_s(f|z)P_t(z|s)} \tag{2}$$

$$P_t(z|s) = \frac{\sum_f v_{ft}P_t(s, z|f)}{\sum_{f,z} v_{ft}P_t(s, z|f)} \tag{3}$$

$$P_t(s) = \frac{\sum_f v_{ft} \sum_{z \in \{\mathbf{z}_s\}} P_t(s, z|f)}{\sum_f v_{ft} \sum_s \sum_{z \in \{\mathbf{z}_s\}} P_t(s, z|f)} \tag{4}$$

The reconstruction of the contribution of source $s$ in the mixture can then be computed as

$$\hat{v}_{ft}^{(s)} = \frac{P_t(s) \sum_{z \in \{\mathbf{z}_s\}} P_s(f|z)P_t(z|s)}{\sum_s P_t(s) \sum_{z \in \{\mathbf{z}_s\}} P_s(f|z)P_t(z|s)} v_{ft}$$

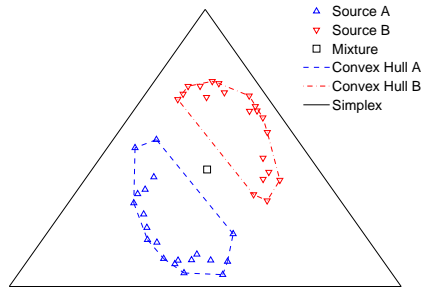

Figure 1: Illustration of the basic model. The triangles denote the position of basis functions for two source classes. The square is an instance of a mixture of the two sources. The mixture point is not within the convex hull which covers either source, but it is within the convex hull defined by all the bases combined.

These reconstructions will approximate the magnitude spectrogram of each source in the mixture. Once we obtain these reconstructions we can use them to modulate the original phase spectrogram of the mixture and obtain the time-series representation of the sources.

Let us now pursue a brief pictorial understanding of this algorithm, which will help us introduce the concepts in the next section. Each basis vector and the mixture input will lie in a $F-1$ dimensional simplex (due to the fact that these quantities are normalized to sum to unity). Each source's basis set will define a convex hull within which any point can be approximated using these bases. Assuming that the training data is accurate, all potential inputs from that source should lie in that area. The union of all the source bases will define a larger space in which a mixture input will be inside. Any mixture point can then be approximated as a weighted sum of multiple bases from both sources. For visualization of these concepts for $F = 3$, see figure 1.

## 2.2 Using Training Data Directly as a Dictionary

In this paper, we would like to explain the mixture frame from the training spectral frames instead of using a smaller set of learned bases. There are two rationales behind this decision. The first is that the resulting large dictionary provides a better description of the sources, as opposed to the less expressive learned-basis models. As we show later on, this holds even for learned-basis models with dictionaries as large as the proposed method's. The secondary rationale behind this operation is based on the observation that the points defined by the convex hull of a source's model, do not necessarily all fall on that source's manifold. To visualize this problem consider the plots in figure 2. In both of these plots the sources exhibit a clear structure. In the left plot both sources appear in a circular pattern, and in the right plot in a spiral form. As shown in [12], learning a set of bases that explains these sources results in defining a convex hull that surrounds the training data. Under this model potential source estimates can now lie anywhere inside these hulls. Using trained-basis models, if we decompose the mixture points in these figures we obtain two source estimates which do not lie in the same manifold as the original sources. Although the input was adequately approximated, there is no guarantee that the extracted sources are indeed appropriate outcomes for their sound class.

In order to address this problem and to also provide a richer dictionary for the source reconstructions, we will make direct use of the training data in order to explain the mixture, and bypass the basis representation as an abstraction. To do so we will use each frame of the spectrograms of the training sequences as the bases $P_s(f|z)$. More specifically, let $\mathbf{W}^{(s)}_{F \times T^{(s)}}$ be the training spectrogram from source $s$ and let $\mathbf{w}^{(s)}_t$ represent the time frame $t$ from the spectrogram. In this case, the latent variable $z$ for source $s$ takes $T^{(s)}$ values, and the $z$-th basis function will be given by the (normalized) $z$-th column vector of $\mathbf{W}^{(s)}$.

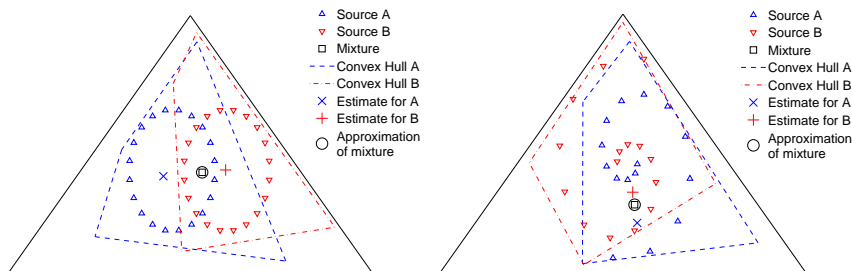

Figure 2: Two examples where the separation process using trained bases provides poor source estimates. In both plots the training data for each source are denoted by △ and ▽, and the mixture sample by □. The learned bases of each source are the vertices of the two dashed convex hulls that enclose each class. The source estimates and the approximation of the mixture are denoted by ×, + and ◯. In the left case the two sources lie on two overlapping circular areas, the source estimates however lie outside these areas. On the right, the two sources form two intertwined spirals. The recovered sources lie very closely on the competing source's area, thereby providing a highly inappropriate decomposition. Although the mixture was well approximated in both cases, the estimated sources were poor representations of their classes.

With the above model we would ideally want to use one dictionary element per source at any point in time. Doing so will ensure that the outputs would lie on the source manifold, and also offset any issues of potential overcompleteness. One way to ensure this is to perform a reconstruction such that we only use one element of each source at any time, much akin to a nearest-neighbor model, albeit in an additive setting. This kind of search can be computationally very demanding so we instead treat this as a sparse approximation problem. The intuition is that at any given point in time, the mixture frame is explained by very few active elements from the training data. In other words, we need the mixture weight distributions and the speaker priors to be sparse at every time instant.

We use the concept of *entropic prior* introduced in [13] to enforce sparsity. Given a probability distribution $\boldsymbol{\theta}$, entropic prior is defined as

$$P_e(\boldsymbol{\theta}) = e^{-\mathcal{H}(\boldsymbol{\theta})} \tag{5}$$

where $\mathcal{H}(\boldsymbol{\theta}) = -\sum_i \theta_i \log \theta_i$ is the entropy of the distribution. A sparse representation, by definition, has few "active" elements which means that the representation has low entropy. Hence, imposing this prior during *maximum a posteriori* estimation is a way to minimize entropy during estimation which will result in a sparse $\boldsymbol{\theta}$ distribution. We would like to minimize the entropies of both the speaker dependent mixture weight distributions (given by $P_t(z|s)$) and the source priors (given by $P_t(s)$) at every frame. In other words, we want to minimize $\mathcal{H}(z|s)$ and $\mathcal{H}(s)$ at every time frame. However, we know from information theory that

$$\mathcal{H}(z,s) = \mathcal{H}(z|s) + \mathcal{H}(s).$$

Thus, reducing the entropy of the joint distribution $P_t(z,s)$ is equivalent to reducing the conditional entropy of the source dependent mixture weights and the entropy of the source priors.

Since the dictionary is already known and is given by the normalized spectral frames from source training spectrograms, the parameter to be estimated is given by $P_t(z,s)$. The model, written in terms of this parameter, is given by

$$P_t(f) = \sum_s \sum_{z \in \{\mathbf{z}_s\}} P_s(f|z) P_t(z,s).$$

where we have modified equation (1) by representing $P_t(s)P_t(z|s)$ as $P_t(z,s)$. We use the Expectation-Maximization algorithm to derive the update equations. Let all parameters to be estimated be represented by $\Lambda$. We impose an entropic prior distribution on $P_t(z,s)$

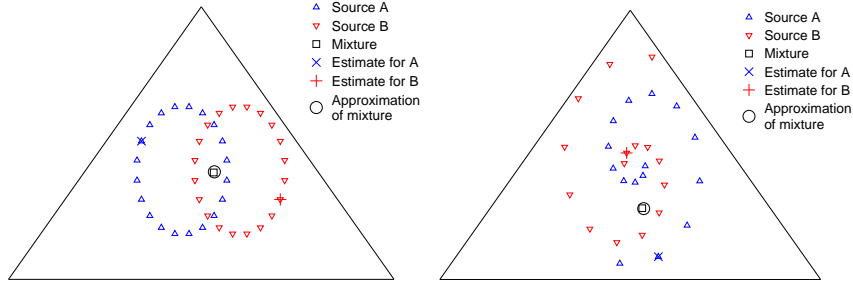

Figure 3: Using a sparse reconstruction on the data in figure 2. Note how in contrast to that figure the source estimates are now identified as training data points, and are thus plausible solutions. The approximation of the mixture is the nearest point of the line connecting the two source estimates, to the actual mixture input. Note that the proper solution is the one that results in such a line that is as close as possible to the mixture point, and not one that is defined by two training points close to the mixture.

given by

$$\log P(\Lambda) = \beta \sum_t \sum_s \sum_{z \in \{\mathbf{z}_s\}} P_t(z,s) \log P_t(z,s),$$

where $\beta$ is a parameter indicating the extent of sparsity desired. The E-step is given by

$$P_t(z,s|f) = \frac{P_t(z,s)P_s(f|z)}{\sum_s \sum_{z \in \{\mathbf{z}_s\}} P_t(z,s)P_s(f|z)}$$

and the M-step by

$$\frac{\omega_t}{P_t(z,s)} + \beta + \beta \log P_t(z,s) + \lambda_t = 0 \tag{6}$$

where we have let $\omega$ represent $\sum_f v_{ft} P_t(s,z|f)$ and $\lambda_t$ is the Lagrange multiplier. The above M-step equation is a system of simultaneous transcendental equations for $P_t(z,s)$. Brand [13] proposes a method to solve such problems using the Lambert $\mathcal{W}$ function [14]. It can be shown that $P_t(z,s)$ can be estimated as

$$\hat{P}_t(z,s) = \frac{-\omega/\beta}{\mathcal{W}(-\omega e^{1+\lambda_t/\beta}/\beta)}. \tag{7}$$

Equations (6),(7) form a set of fixed point iterations that typically converge in 2-5 iterations [13].

Once $P_t(z,s)$ is estimated, the reconstruction of source $s$ can be computed as

$$\hat{v}_{ft}^{(s)} = \frac{\sum_{z \in \{\mathbf{z}_s\}} P_s(f|z)P_t(z,s)}{\sum_s \sum_{z \in \{\mathbf{z}_s\}} P_s(f|z)P_t(z,s)} v_{ft}$$

Now let us consider how this problem resolves the issues presented in figure 2. In figure 3 we show the results obtained using this approach on the same data. The sparsity parameter $\beta$ as set to 0.1. In both plots we see that the source reconstructions lie on a training point, thereby being a plausible source estimate. The approximation of the mixture is not as exact as before, since now it has to lie on the line connecting the two active source elements. This is not however an issue of concern since in practice the approximation is always good enough, and the guarantee of a plausible source estimate is more valuable than the exact approximation of the mixture.

Alternative means to strive towards similar results would be to make use of priors such as in [15, 16]. In these approaches the priors are imposed on the mixture weights and thus are not as effective for this particular task since they still suffer from the symptoms of learned-basis models. This was verified through cursory simulations, which also revealed an additional computational complexity penalty against such models.

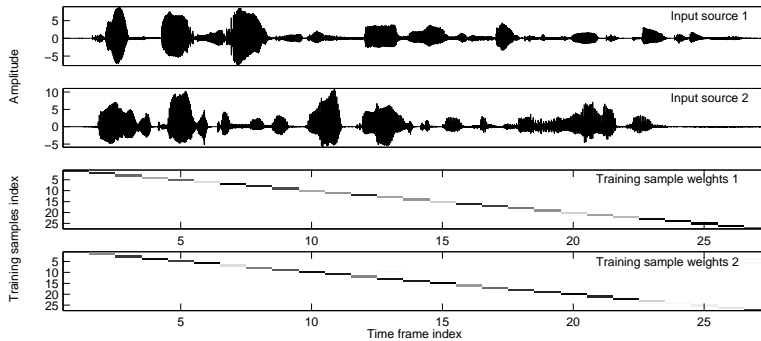

Figure 4: An oracle case where we fit training data from two speakers, on the mixture of that data. The top plots show the input waveforms, and the bottom plots shows the estimated weights multiplied with the source priors. As expected the weights exhibit two diagonal traces which imply that the algorithm we used has fit the data appropriately.

## 3    Experimental Results

In this section we present the results of experiments done with real speech data. All of these experiments we performed on data from the TIMIT speech database on 0dB male/female mixtures. The sources were sampled as 16 kHz, we used 64 ms windows for the spectrogram computation, and an overlap of 32 ms. Before the FFT computation, the input was tapered using a square-root Hann window. The training data was around 25 sec worth of speech for each speaker, and the testing mixture was about 3 sec long. We evaluated the separation performance using the metrics provided in [17]. These metrics include the Signal to Interference Ratio (SIR), the Signal to Distortion Ratio (SDR), and the Signal to Artifacts Ratio (SAR). The first is a measure of how well we suppress the interfering speaker, whereas the other two provide us with a sense of how much the extracted source is corrupted due to the separation process. All of these are measured in dB and the higher they are the better the performance is deemed to be.

In the following sections we first present some "oracle tests" that validate that indeed this algorithm is performing as expected, and we then proceed to more realistic testing. Finally, we show the performance impact of pruning the training data in order to speed up computation time.

### 3.1    Oracle tests

In order to verify that this approach works we go through a few oracle experiments. In these tests we include the actual solutions as training data and we make sure that the answers are exactly what we would expect to find. The first experiment we perform is on a mixture for which the training data includes its isolated constituent sentences. In this experiment we would expect to see two dictionary components active at each point in time, one from each speaker's dictionary, and both of these progressing through the component index linearly through time. As shown in figure 4, we observe exactly that behavior. This test provides a sanity check which verifies that given an answer this algorithm can properly identify it.

A more comprehensive oracle test is shown in figure 5. In this experiment, the training data were again the same as the testing data. We averaged the results from 10 runs using different combinations of speakers, varying sparsity parameters and number of bases. The sparsity parameter $\beta$ was checked for various values from 0 to 0.8, and we used trained-basis models with 5, 10, 20, 40, 80, 160 and 320 bases, as well as the proposed scenario where all the training data is used as a dictionary. The primary observation from this experiment is that the more bases we use the better the results get. We also see that increasing the sparsity parameter we see a modest improvement in most cases.

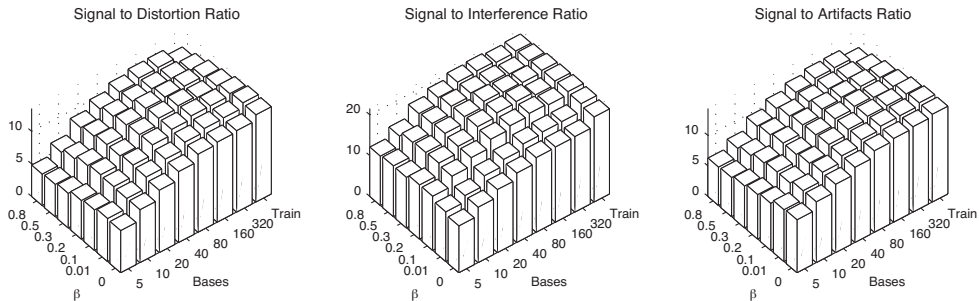

Figure 5: Average separation performance metrics for oracle cases, as dependent on the choice of different number of elements in the speaker's dictionary, and different choices of the entropic prior parameter $\beta$. The left plot shows the SDR, the middle plot the SIR, and the right plot the SAR, all in dB. The basis row labeled as "Train" is the case where we use all the training data as a basis set.

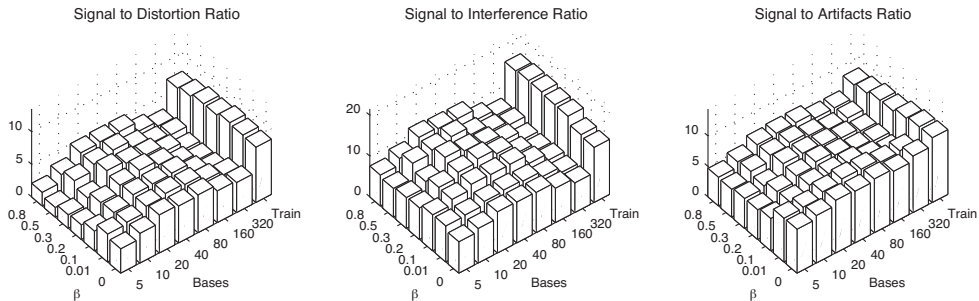

Figure 6: Average separation performance metrics for real-world cases, as dependent on the choice of different number of elements in the speaker's dictionary, and different choices of the entropic prior parameter $\beta$. The left plot shows the SDR, the middle plot the SIR, and the right plot the SAR, all in dB. Sparsely using all of the training data clearly outperforms low-rank models by a significant margin on all metrics.

## 3.2    Results on Realistic Situations

Let us now consider the more realistic case where the mixture data is different from the training set. In the following simulation we repeat the previous experiment, but in this case there are no common elements between the training and testing data. The input mixture has to be reconstructed using approximate samples. The results are now very different in nature. We do not obtain such high numbers in performance as in the oracle case, but we also see a stronger trend in favor of sparsity and the use of all the training data as a dictionary. The results are shown in figure 6. We can clearly see that in all metrics using all the training data significantly outperforms trained-basis models. More importantly, we see that this is not because we have a larger dictionary. For trained-bases we see a performance peak at around 80 bases, but then we observe a deterioration in performance as we use a larger dictionary. Using the actual training data results in a significant boost though. Due to the high dimensionality of the data the effect of sparsity is a little more subtle, but we still see a helpful boost especially for the SIR which is the most important of the performance measures. We see some decrease in the SAR, which is expected since the reconstructions are made using elements that look like the remaining data, and are not made to approximate the actual input mixture. This does not mean that the extracted sources are distorted and of poor quality, but rather that they don't match the original inputs exactly. The use of sparsity ensures that the output is a plausible speech signal devoid of artifacts like distortion and musical noise. The effects of sparsity alone in the proposed case are shown separately in figure 7.

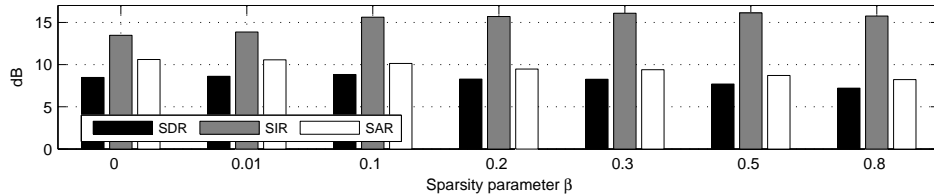

Figure 7: A slice of the results in figure 6 in which we only show the case where we use all the training data as adictionary. The horizontal axis represents various values for the sparsity parameter $\beta$.

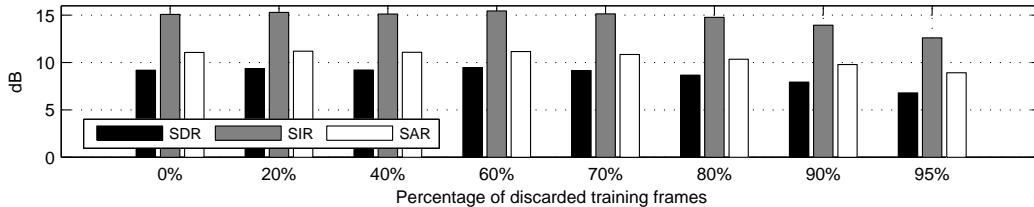

Figure 8: Effect of discarding low energy training frames. The horizontal axis denotes the percentage of training frames that have been discarded. These are averaged results using a sparsity parameter $\beta = 0.1$.

The unfortunate side effect of the proposed method is that we need to use a dictionary which can be substantially larger than otherwise. In order to address this concern we show that the size of the training data can be easily pruned down to a size comparable to trained-basis models and still outperform them. Since sound signals, especially speech, tend to have a considerable amount of short-term pauses and regions of silence, we can use an energy threshold to in order to select the loudest frames of the training spectrogram as bases. In figure 8 we show how the separation performance metrics are influenced as we increasingly remove bases which lie under various energy percentiles. It is clear that even after discarding up to at least 70% of the lowest energy training frames the performance is still approximately the same. After that we see some degradation since we start discarding significant parts of the training data. Regardless this scheme outperforms trained-basis models of equivalent size. For the 80% percentile case, a trained-basis model of the same size dictionary results in roughly half the values in all performance metrics, a very significant handicap for the same amount of computational and memory requirements.

The experiments in this paper were all conducted in MATLAB on an average modern desktop machine. Overall computations for a single mixture took roughly 4 sec when not using the sparsity prior, 14 sec when using the sparsity prior (primarily due to slow computation of Lambert's function), and dropped down to 5 sec when using the 30% highest energy frames from the training data.

## 4   Conclusion

In this paper we present a new approach to solving the monophonic source separation problem. The contributions of this paper lies primarily in the choice of using all the training data as opposed to a trained-basis model. In order to do so we present a sparse learning algorithm which can efficiently solve this problem, and also guarantees that the returned source estimates are plausible given the training data. We provide experiments that show how this approach is influenced by the use of varying sparsity constraints and training data selection. Finally we demonstrate how this approach can generate significantly superior results as compared to trained-basis methods.

# References

[1] S. T. Roweis, One microphone source separation, in Advances in Neural Information Processing Systems, 2001.

[2] Reddy, A.M. and B. Raj. Soft Mask Methods for Single-Channel Speaker Separation, in IEEE Transactions of Audio, Speech, and Language Processing, Volume: 15, Issue: 6, Aug 2007.

[3] T. Kristjansson, J. Hershey, P. Olsen, S. Rennie, and R. Gopinath, Super-human multi-talker speech recognition: The IBM 2006 speech separation challenge system, in International Conference on Spoken Language Processing (INTERSPEECH), 2006, pp. 97–100, Kluwer Academic Publishers, ch. 20, pp. 295304.

[4] Casey, M.A., and A. Westner. Separation of mixed audio sources by independent sub-space analysis, in Proceedings of the International Conference of Computer Music, 2000.

[5] Jang, G.-J., T.-W. Lee. A Maximum Likelihood Approach to Single-channel Source Separation, in Journal of Machine Learning Research 4 (2003) pp. 1365–1392.

[6] Pearlmutter, B., M. Zibulevsky, Blind Source Separation by Sparse Decomposition in a Signal Dictionary, in Neural Computation 13, pp. 863–882. 2001.

[7] L. Benaroya, L. M. Donagh, F. Bimbot, and R. Gribonval, Non negative sparse representation for wiener based source separation with a single sensor, in Acoustics, Speech, and Signal Processing, IEEE International Conference on, 2003, pp. 613–616.

[8] M. N. Schmidt and R. K. Olsson, Single-channel speech separation using sparse non-negative matrix factorization, in International Conference on Spoken Language Processing (INTERSPEECH), 2006.

[9] T. Virtanen, Sound source separation using sparse coding with temporal continuity objective, in International Computer Music Conference, ICMC, 2003.

[10] Smaragdis, P. Raj, B. and Shashanka, M.V. 2007. Supervised and Semi-Supervised Separation of Sounds from Single-Channel Mixtures. In proceedings of ICA 2007. London, UK. September 2007.

[11] Raj, B.; Smaragdis, P. 2005. Latent Variable Decomposition of Spectrograms for single channel speaker separation. In Proceedings of the IEEE Workshop on Applications of Signal Processing to Audio and Acoustics, New Paltz, NY, October, 2005.

[12] Shashanka, M.V., B. Raj, P. Smaragdis, 2007. Sparse Overcomplete Latent Variable Decoposition of Counts Data. In Neural Information Processing Systems (NIPS), Vancouver, BC, Canada. December 2007.

[13] Brand, M.E. Pattern Discovery via Entropy Minimization. In Uncertainty 99, AISTATS99,1999.

[14] Corless, R.M., G.H. Gonnet, D.E.G. Hare, D.J. Jeffrey, and D.E. Knuth. On the Lambert W Function. Advances in Computational Mathematics,1996.

[15] Bouguila N. and D. Ziou. Using unsupervised learning of a finite Dirichlet mixture model to improve pattern recognition applications, Pattern Recognition Letters, Volume 26, Issue 12, September 2005.

[16] Hinneburg, A., Gabriel, H.-H. and Gohr, A. Bayesian Folding-In with Dirichlet Kernels for PLSI, in Seventh IEEE International Conference on Data Mining, Oct. 2007

[17] Févotte, C., R. Gribonval and E. Vincent. 2005. BSS EVAL Toolbox User Guide, IRISA Technical Report 1706, Rennes, France, April 2005.

